# Plasticity-Mediated Competitive Learning

**Nicol N. Schraudolph**
nici@salk.edu

**Terrence J. Sejnowski**
terry@salk.edu

Computational Neurobiology Laboratory
The Salk Institute for Biological Studies
San Diego, CA 92186-5800

*and*

Computer Science & Engineering Department
University of California, San Diego
La Jolla, CA 92093-0114

## Abstract

Differentiation between the nodes of a competitive learning network is conventionally achieved through competition on the basis of neural activity. Simple inhibitory mechanisms are limited to sparse representations, while decorrelation and factorization schemes that support distributed representations are computationally unattractive. By letting neural *plasticity* mediate the competitive interaction instead, we obtain diffuse, nonadaptive alternatives for fully distributed representations. We use this technique to simplify and improve our binary information gain optimization algorithm for feature extraction (Schraudolph and Sejnowski, 1993); the same approach could be used to improve other learning algorithms.

## 1 INTRODUCTION

Unsupervised neural networks frequently employ sets of nodes or subnetworks with identical architecture and objective function. Some form of competitive interaction is then needed for these nodes to differentiate and efficiently complement each other in their task.

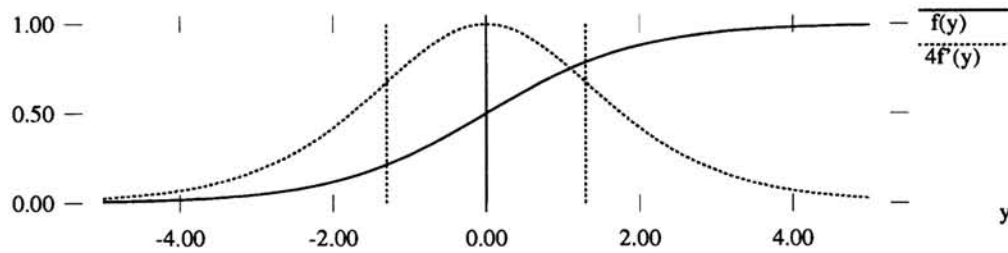

Figure 1: Activity $f$ and plasticity $f'$ of a logistic node as a function of its net input $y$. Vertical lines indicate those values of $y$ whose pre-images in input space are depicted in Figure 2.

Inhibition is the simplest competitive mechanism: the most active nodes suppress the ability of their peers to learn, either directly or by depressing their activity. Since inhibition can be implemented by diffuse, nonadaptive mechanisms, it is an attractive solution from both neurobiological and computational points of view. However, inhibition can only form either localized (unary) or sparse distributed representations, in which each output has only one state with significant information content.

For fully distributed representations, schemes to decorrelate (Barlow and Földiák, 1989; Leen, 1991) and even factorize (Schmidhuber, 1992; Bell and Sejnowski, 1995) node activities do exist. Unfortunately these require specific, weighted lateral connections whose adaptation is computationally expensive and may interfere with feedforward learning. While they certainly have their place as competitive learning algorithms, the capability of biological neurons to implement them seems questionable.

In this paper, we suggest an alternative approach: we extend the advantages of simple inhibition to distributed representations by decoupling the competition from the activation vector. In particular, we use neural *plasticity* — the derivative of a logistic activation function — as a medium for competition.

Plasticity is low for both high and low activation values but high for intermediate ones (Figure 1); distributed patterns of activity may therefore have localized plasticity. If competition is controlled by plasticity then, simple competitive mechanisms will constrain us to localized plasticity but allow representations with distributed activity.

The next section reintroduces the binary information gain optimization (BINGO) algorithm for a single node; we then discuss how plasticity-mediated competition improves upon the decorrelation mechanism used in our original extension to multiple nodes. Finally, we establish a close relationship between the plasticity and the entropy of a logistic node that provides an intuitive interpretation of plasticity-mediated competitive learning in this context.

## 2   BINARY INFORMATION GAIN OPTIMIZATION

In (Schraudolph and Sejnowski, 1993), we proposed an unsupervised learning rule that uses logistic nodes to seek out binary features in its input. The output

$$z = f(y), \text{ where } f(y) = \frac{1}{1 + e^{-y}} \text{ and } y = \vec{w} \cdot \vec{x} \tag{1}$$

of each node is interpreted stochastically as the probability that a given feature is present. We then search for informative directions in weight space by maximizing the information gained about an unknown binary feature through observation of $z$. This *binary information gain* is given by

$$\Delta H(z) = H(\hat{z}) - H(z), \tag{2}$$

where $H(z)$ is the entropy of a binary random variable with probability $z$, and $\hat{z}$ is a prediction of $z$ based on prior knowledge. Gradient ascent in this objective results in the learning rule

$$\Delta \vec{w} \propto f'(y) \cdot (y - \hat{y}) \cdot \vec{x}, \tag{3}$$

where $\hat{y}$ is a prediction of $y$. In the simplest case, $\hat{y}$ is an empirical average $\langle y \rangle$ of past activity, computed either over batches of input data or by means of an exponential trace; this amounts to a nonlinear version of the *covariance rule* (Sejnowski, 1977).

Using just the average as prediction introduces a strong preference for splitting the data into two equal-sized clusters. While such a bias is appropriate in the initial phase of learning, it fails to take the nonlinear nature of $f$ into account. In order to discount data in the saturated regions of the logistic function appropriately, we weigh the average by the node's plasticity $f'(y)$:

$$\hat{y} = \frac{\langle y \cdot f'(y) \rangle}{\langle f'(y) \rangle + \varepsilon}, \tag{4}$$

where $\varepsilon$ is a very small positive constant introduced to ensure numerical stability for large values of $y$. Now the bias for splitting the data evenly is gradually relaxed as the network's weights grow and data begins to fall into saturated regions of $f$.

## 3   PLASTICITY-MEDIATED COMPETITION

For multiple nodes the original BINGO algorithm used a decorrelating predictor as the competitive mechanism:

$$\widehat{\vec{y}} = \vec{y} + (Q_{\vec{y}} - 2I)(\vec{y} - \langle \vec{y} \rangle), \tag{5}$$

where $Q_{\vec{y}}$ is the autocorrelation matrix of $\vec{y}$, and $I$ the identity matrix. Note that $Q_{\vec{y}}$ is computationally expensive to maintain; in connectionist implementations it

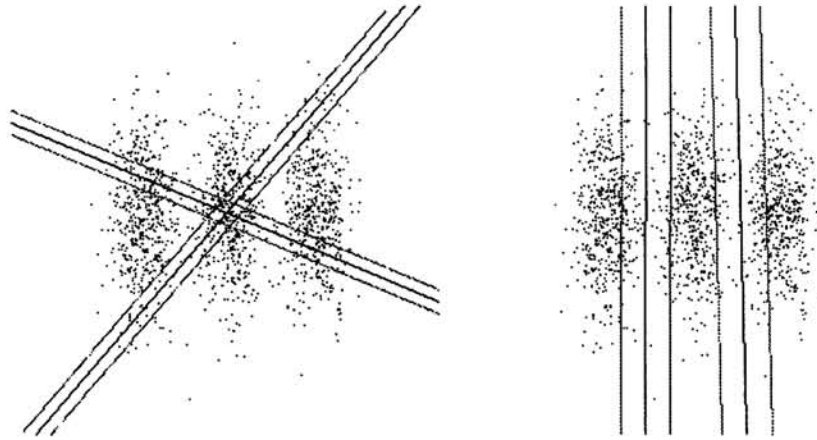

Figure 2: The "three cigars" problem. Each plot shows the pre-image of zero net input, superimposed on a scatter plot of the data set, in input space. The two flanking lines delineate the "plastic region" where the logistic is not saturated, providing an indication of weight vector size. Left, two-node BINGO network using decorrelation (Equations 3 & 5) fails to separate the three data clusters. Right, same network using plasticity-mediated competition (Equations 4 & 6) succeeds.

is often approximated by lateral anti-Hebbian connections whose adaptation must occur on a faster time scale than that of the feedforward weights (Equation 3) for reasons of stability (Leen, 1991). In practice this means that learning is slowed significantly.

In addition, decorrelation can be inappropriate when nonlinear objectives are optimized — in our case, two prominent binary features may well be correlated. Consider the "three cigars" problem illustrated in Figure 2: the decorrelating predictor (left) forces the two nodes into a near-orthogonal arrangement, interfering with their ability to detect the parallel gaps separating the data clusters.

For our purposes, decorrelation is thus too strong a constraint on the discriminants: all we require is that the discovered features be distinct. We achieve this by reverting to the simple predictor of Equation 4 while adding a global, plasticity-mediated excitation[1] factor to the weight update:

$$\Delta \vec{w}_i \propto f'(y_i) \cdot (y_i - \hat{y}_i) \cdot \vec{x} \cdot \sum_j f'(y_j) \qquad (6)$$

As Figure 2 (right) illustrates, this arrangement solves the "three cigars" problem. In the high-dimensional environment of hand-written digit recognition, this algorithm discovers a set of distributed binary features that preserve most of the information needed to classify the digits, even though the network was never given any class labels (Figure 3).

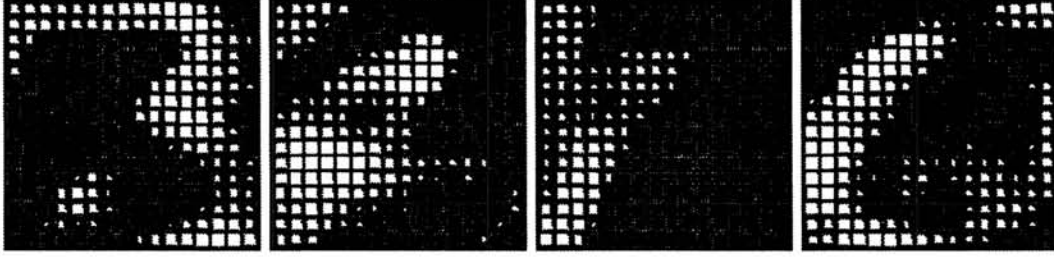

Figure 3: Weights found by a four-node network running the improved BINGO algorithm (Equations 4 & 6) on a set of 1200 handwritten digits due to (Guyon et al., 1989). Although the network is unsupervised, its four-bit output conveys most of the information necessary to classify the digits.

## 4 PLASTICITY AND BINARY ENTROPY

It is possible to establish a relationship between the plasticity $f'$ of a logistic node and its entropy that provides an intuitive account of plasticity-mediated competition as applied to BINGO. Consider the binary entropy

$$H(z) = -z \log z - (1 - z) \log(1 - z) \tag{7}$$

A well-known quadratic approximation is

$$\tilde{H}(z) = 8e^{-1} z (1 - z) \approx H(z) \tag{8}$$

Now observe that the plasticity of a logistic node

$$f'(y) = \frac{\partial}{\partial y} \frac{1}{1 + e^{-y}} = \ldots = z(1 - z) \tag{9}$$

is in fact proportional to $\tilde{H}(z)$ — that is, a logistic node's plasticity is in effect a convenient quadratic approximation to its binary output entropy. The overall entropy in a layer of such nodes equals the sum of individual entropies less their redundancy:

$$H(\vec{z}) = \sum_j H(z_j) - R(\vec{z}) \tag{10}$$

The plasticity-mediated excitation factor in Equation 6

$$\sum_j f'(y_j) \propto \sum_j \tilde{H}(z_j), \tag{11}$$

is thus proportional to an approximate upper bound on the entropy of the layer, which in turn indicates how much more information remains to be gained by learning from a particular input. In the context of BINGO, plasticity-mediated

competition thus scales weight changes according to a measure of the network's ignorance: the less it is able to identify a given input in terms of its set of binary features, the more it tries to learn doing so.

## 5  CONCLUSION

By using the derivative of a logistic activation function as a medium for competitive interaction, we were able to obtain differentiated, fully distributed representations without resorting to computationally expensive decorrelation schemes. We have demonstrated this plasticity-mediated competition approach on the BINGO feature extraction algorithm, which is significantly improved by it. A close relationship between the plasticity of a logistic node and its binary output entropy provides an intuitive interpretation of this unusual form of competition.

Our general approach of using a nonmonotonic function of activity — rather than activity itself — to control competitive interactions may prove valuable in other learning schemes, in particular those that seek distributed rather than local representations.

### Acknowledgements

We thank Rich Zemel and Paul Viola for stimulating discussions, and the Mc-Donnell-Pew Center for Cognitive Neuroscience in San Diego for financial support.

## Footnotes

[1]The interaction is excitatory rather than inhibitory since a node's plasticity is *inversely* correlated with the magnitude of its net input.

### References

Barlow, H. B. and Földiák, P. (1989). Adaptation and decorrelation in the cortex. In Durbin, R. M., Miall, C., and Mitchison, G. J., editors, *The Computing Neuron*, chapter 4, pages 54–72. Addison-Wesley, Wokingham.

Bell, A. J. and Sejnowski, T. J. (1995). A non-linear information maximisation algorithm that performs blind separation. In *Advances in Neural Information Processing Systems*, volume 7, Denver 1994.

Guyon, I., Poujaud, I., Personnaz, L., Dreyfus, G., Denker, J., and Le Cun, Y. (1989). Comparing different neural network architectures for classifying handwritten digits. In *Proceedings of the International Joint Conference on Neural Networks*, volume II, pages 127–132. IEEE.

Leen, T. K. (1991). Dynamics of learning in linear feature-discovery networks. *Network*, 2:85–105.

Schmidhuber, J. (1992). Learning factorial codes by predictability minimization. *Neural Computation*, 4(6):863–879.

Schraudolph, N. N. and Sejnowski, T. J. (1993). Unsupervised discrimination of clustered data via optimization of binary information gain. In Hanson, S. J., Cowan, J. D., and Giles, C. L., editors, *Advances in Neural Information Processing Systems*, volume 5, pages 499–506, Denver 1992. Morgan Kaufmann, San Mateo.

Sejnowski, T. J. (1977). Storing covariance with nonlinearly interacting neurons. *Journal of Mathematical Biology*, 4:303–321.